# A V1 model of pop out and asymmetry in visual search

**Zhaoping Li**
University College London, z.li@ucl.ac.uk

## Abstract

Visual search is the task of finding a target in an image against a background of distractors. Unique features of targets enable them to pop out against the background, while targets defined by lacks of features or conjunctions of features are more difficult to spot. It is known that the ease of target detection can change when the roles of figure and ground are switched. The mechanisms underlying the ease of pop out and asymmetry in visual search have been elusive. This paper shows that a model of segmentation in V1 based on intracortical interactions can explain many of the qualitative aspects of visual search.

## 1 Introduction

Visual search is closely related to visual segmentation, and therefore can be used to diagnose the mechanisms of visual segmentation. For instance, a red dot can pop-out against a background of green distractor dots instantaneously, suggesting that only pre-attentive mechanisms are necessary (Treisman et al, 1990). On the other hand, it is much more difficult to search for a red 'X' among green 'X's and red 'O's – the time it takes to detect the target's presence increases with the number of background distractors, suggesting some form of attentive serial search. Sometimes, the search times change when the role of the figure (target) and ground (distractors) are switched -- *asymmetry* in visual search. For instance, it is easier to find a longer bar in a background of shorter bars than vice-versa.

It has been unclear which visual areas or neural mechanisms are responsible for the pop out and asymmetry in visual search. There are, however, psychophysical theories (Treisman et al 1990, Treisman and Gormican 1988) which argue that visual inputs are coded in a number of primitive or basic feature dimensions: orientation, color, brightness, motion direction, disparity, line ends, line intersections, and closure. A target can pop-out preattentively if it has a feature in one of these dimensions, such as a particular color or orientation, which is absent in the distrac-

tors. Hence, a red dot pops out among green ones. However, red 'X' is difficult to spot among green 'X's and red 'O's because neither being red nor being 'X' is unique for the target, and therefore serial search is required. While a vertical line pops out of horizontal ones and vice versa without any search asymmetry, search asymmetry will arise when a single feature in which target and distractors differ is present in one of the two and absent or reduced in the other. Hence, a long line is more easily spotted among short lines than the reserve. This theory has been very helpful in understanding search phenomena. However, it has to make assumptions about what are the primitive feature dimensions, as well as what constitutes larger or smaller values along a given dimension. For instance, to explain that a curved line is more easily spotted among straight lines than the reverse, the theory has to define straightness as the default or standard, and curvaciousness as the deviation from this standard and thus an added feature. Empirically, other pairs of standard and deviant properties include vertical versus tilted, parallel versus convergent, short vs long lines, circle vs ellipse, and complete versus incomplete circles. The basis behind these assumptions are not completely clear. Other related theories have similar problems. For instance, Julesz's texton theory (Julesz 1981) for visual segmentation or pop out starts off by assuming a complete set of special features that constitute textons.

This paper proposes and demonstrates in a model that pre-attentive mechanisms in V1 can qualitatively explain many of the phenomena of visual search. It is assumed that the ease of search is determined by the relative saliencies of the target and distractors. Intracortical interactions in V1 alter the saliencies of targets and distractors according to their own image features as well as those of the distractor or targets images that form the context. Hence, the relative saliency depends on the particular target-distractor pair involved. In particular, asymmetry is a natural consequence of contextual influences.

## 2 The V1 model

We use a V1 model of pre-attentive visual segmentation which has been shown to be able to detect and highlight smooth contours in noisy backgrounds and find boundaries between texture regions in images (Li 1998a, 1998b). Its behavior agrees with physiological observations (Knierim and van Essen 1992, Kapadia et al 1995). Without loss of generality, the model ignores color, motion, and stereo dimensions, includes mainly layer 2-3 orientation selective cells, and ignores the intra-hypercolumnar mechanism by which their receptive fields are formed. Inputs to the model are images filtered by the edge- or bar-like local receptive fields (RFs) of V1 cells.[1] The cells influence each other contextually via horizontal intra-cortical connections (Rockland and Lund 1983, Gilbert, 1992), transforming patterns of inputs to patterns of cell responses. Fig. 1 shows the elements of the model and their interactions. At each location $i$ there is a model V1 hypercolumn composed of $K$ neuron pairs. Each pair $(i, \theta)$ has RF center $i$ and preferred orientation $\theta = k\pi/K$ for $k = 1, 2, ...K$, and is called (the neural representation of) an edge segment. Based on experimental data (White, 1989, Douglas and Martin 1990), each edge segment consists of an excitatory and an inhibitory neuron that are interconnected, and each model cell represents a collection of local cells of similar types. The excitatory cell receives the visual input; its output is used as a measure of the response or salience of the edge segment and projects to higher visual areas. The inhibitory cells are treated as interneurons. Based on observations by Gilbert, Lund and their colleagues (Rockland and Lund, 1983, Gilbert 1992) horizontal connections $J_{i\theta,j\theta'}$

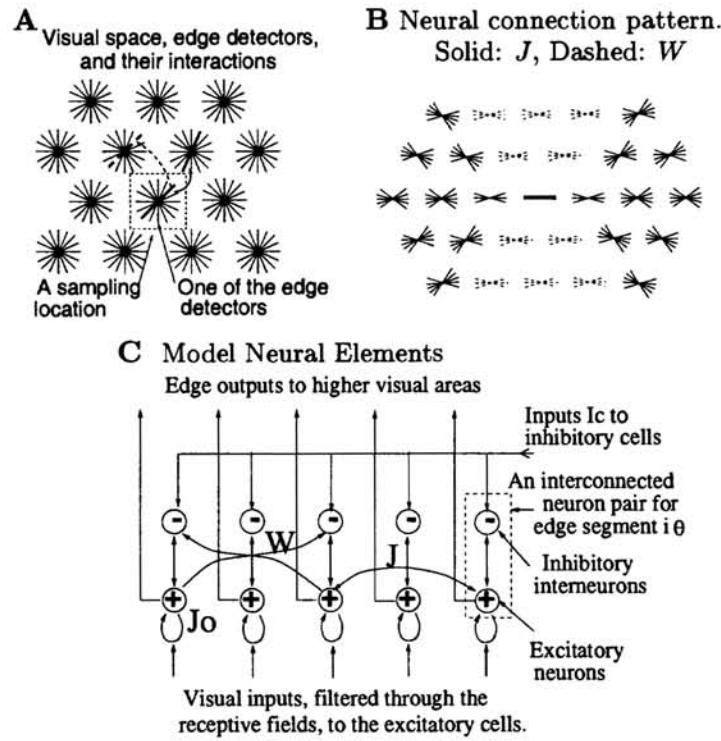

**A** Visual space, edge detectors, and their interactions

A sampling location

One of the edge detectors

**B** Neural connection pattern.
Solid: $J$, Dashed: $W$

**C** Model Neural Elements

Edge outputs to higher visual areas

Inputs Ic to inhibitory cells

An interconnected neuron pair for edge segment i θ

Inhibitory interneurons

Excitatory neurons

Visual inputs, filtered through the receptive fields, to the excitatory cells.

Figure 1: **A:** Visual inputs are sampled in a discrete grid of edge/bar detectors. Each grid point $i$ has $K$ neuron pairs (see **C**), one per bar segment, tuned to different orientations $\theta$ spanning $180°$. Two segments at different grid points can interact with each other via monosynaptic excitation $J$ (the solid arrow from one thick bar to anothe r) or disynaptic inhibition $W$ (the dashed arrow to a thick dashed bar). See also **C**. **B:** A schematic of the neural connection pattern from the center (thick solid) bar to neighboring bars within a few sampling unit distances. $J$'s contacts are shown by thin solid bars. $W$'s are shown by thin dashed bars. The connection pattern is translation and rotation invariant. **C:** An input bar segment is directly processed by an interconnected pair of excitatory and inhibitory cells, each cell models abstractly a local group of cells of the same type. The excitatory cell receives visual input and sends output $g_x(x_{i\theta})$ to higher centers. The inhibitory cell is an interneuron. Visual space is taken as having periodic boundary conditions.

(respectively $W_{i\theta,j\theta'}$) mediate contextual influences via monosynaptic excitation (respectively disynaptic inhibition) from $j\theta'$ to $i\theta$ which have nearby but different RF centers, $i \neq j$, and similar orientation preferences, $\theta \sim \theta'$. The membrane potentials follow the equations:

$$\dot{x}_{i\theta} = -\alpha_x x_{i\theta} - \sum_{\Delta\theta}\psi(\Delta\theta)g_y(y_{i,\theta+\Delta\theta}) + J_o g_x(x_{i\theta}) + \sum_{j\neq i,\theta'}J_{i\theta,j\theta'}g_x(x_{j\theta'}) + I_{i\theta} + I_o$$

$$\dot{y}_{i\theta} = -\alpha_y y_{i\theta} + g_x(x_{i\theta}) + \sum_{j\neq i,\theta'}W_{i\theta,j\theta'}g_x(x_{j\theta'}) + I_c$$

where $\alpha_x x_{i\theta}$ and $\alpha_y y_{i\theta}$ model the decay to resting potential, $g_x(x)$ and $g_y(y)$ are sigmoid-like functions modeling cells' firing rates in response to membrane potentials $x$ and $y$, respectively, $\psi(\Delta\theta)$ is the spread of inhibition within a hypercolumn, $J_o g_x(x_{i\theta})$ is self excitation, $I_c$ and $I_o$ are background inputs, including noise and inputs modeling the general and local normalization of activities (see Li (1998b) for more details). Visual input $I_{i\theta}$ persists after onset, and initializes the activity levels $g_x(x_{i\theta})$. The activities are then modified by the contextual influences. Depending on the visual input, the system often settles into an oscillatory state (Gray

and Singer, 1989, see the details in Li 1998b). Temporal averages of $g_x(x_{i\theta})$ over several oscillation cycles are used as the model's output. The nature of the computation performed by the model is determined largely by the horizontal connections $J$ and $W$, which are local (spanning only a few hypercolumns), and translation and rotation invariant (Fig. 1B).

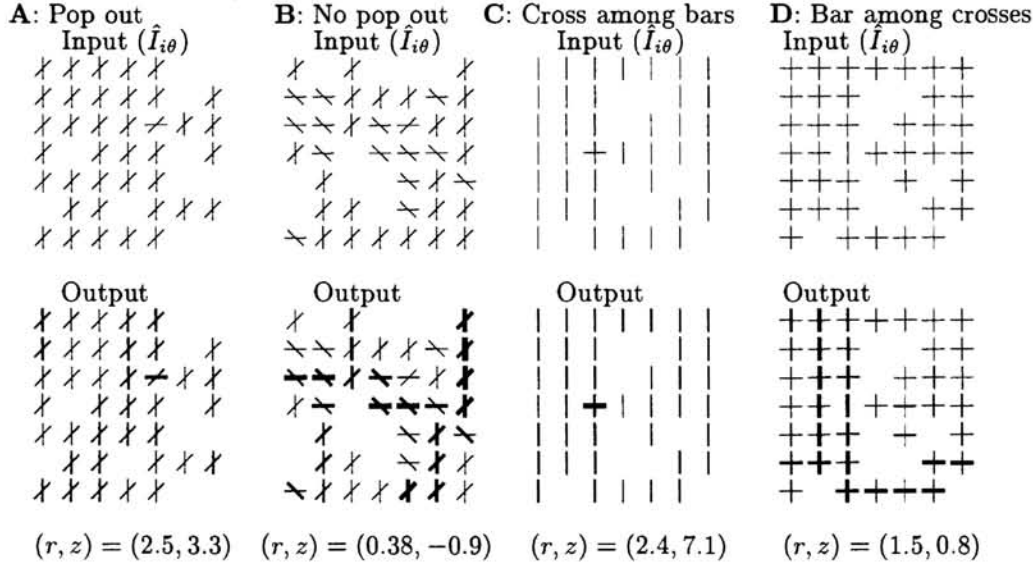

$$(r,z) = (2.5, 3.3) \quad (r,z) = (0.38, -0.9) \quad (r,z) = (2.4, 7.1) \quad (r,z) = (1.5, 0.8)$$

Figure 2: Visual search examples plotted by the model inputs and outputs. A: A single distinctive feature, the horizontal bar in the target, enables pop out. This target is the most salient (measured as the saliency of the horizontal bar in target) spot in the image. B: The target does not pop out since neither of its features, a horizontal and a 45° bars, is unique in the image. The target is less salient than average in the image. C and D demonstrate the asymmetry in a target-distractor pair. C: The cross is the most salient (measured by the saliency of the horizontal bar) spot in the image. The popout strength is stronger than in A. D: The target bar does not pop out,

The model was applied to a variety of input patterns, as shown in examples in the figures. The input values $\hat{I}_{i\theta}$ are the same for all visible bars in each example. The differences in the outputs are caused by intracortical interactions. They become significant about one membrane time constant after the initial neural response (Li, 1998b). The widths of the bars in the figures are proportional to input and output strengths. The plotted region in each picture is often a small region of an extended image. The same model parameters (*e.g.* the dependence of the synaptic weights on distances and orientations, the thresholds and gains in the functions $g_x()$ and $g_y()$, and the level of input noise in $I_o$) are used for all the simulation examples.

We define the net saliency $S_i$ at each grid point $i$ as that of the most activated bar. Define $\bar{S}$ and $\sigma_s$ be the mean and standard deviation of the saliencies of all grid points with visible stimuli. Let $r_i \equiv S_i/\bar{S}$ and $z_i \equiv (S_i - \bar{S})/\sigma_s$. A highly salient point $i$ should have large values of $(r_i, z_i)$ – in particular, both $r_i$ and $z_i$ should be larger than 1. For larger targets that occupy more than one grid point, the relative saliency measure of the target is that of the most salient grid point on the target.

Fig. (2)A,B compare the state of the target '⟋' in two different contexts. Against a texture of '⟋' it is highly salient because of its unique horizontal bar. Against '⟋' and '⟍' it is much less salient because only the conjunction of '—' and '⟋' distinguishes it. Fig. (2)C,D exhibit search asymmetry. The horizontal bar in the target is unique in the image of Fig. (2)A,C, which leads to pop out, and each target sits at the most salient location in the respective images. On the other hand, no feature in the targets of Fig. (2)B,D is unique. These examples are consistent with the psychophysical

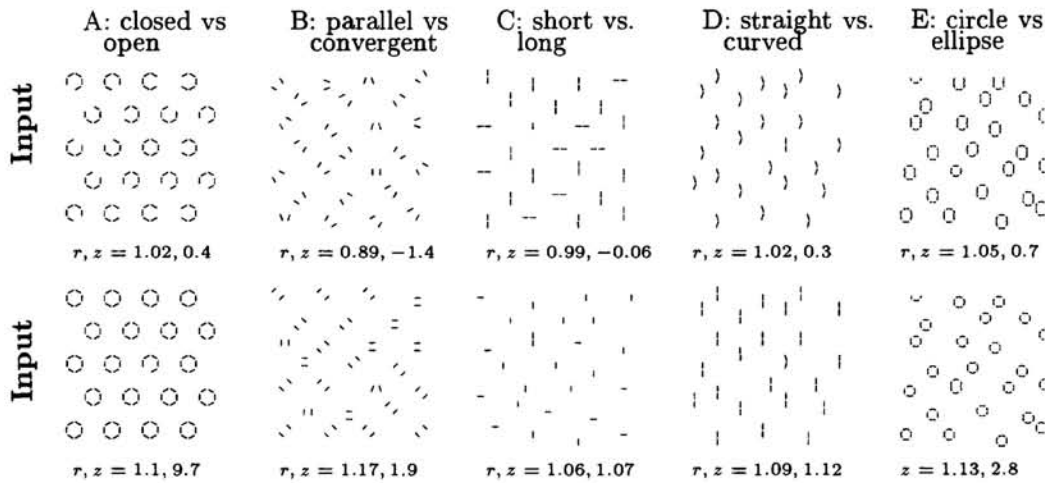

Figure 3: Five typical examples, one column each, of visual search asymmetry as simulated in the model. The input stimuli are plotted, the target saliency $r, z$ scores are indicated below each of them. All input bars are of the same intermediate input contrast. The role of figure and ground is switched from the top to the bottom rows.

theories mentioned in introduction. Further, we note that because intracortical interactions link mostly neurons preferring similar orientations, two very different orientations can be viewed as independent features. The pop out is stronger in Fig. (2)C than Fig. (2)A since horizontal differs more from vertical ($90°$) than from $45°$. The V1 orientation selective RFs and orientation specific horizontal connnections provide the neural basis for orientation as one of the primitive feature dimensions. In fact, the contextual influences between image features imply that saliency values depend on detailed geometrical relationships between features within and between a target or distrator and its nearby target or distractors (see Fig. (2)B). The relative ease in searches varies continuously from extreme pop out to slow serial searches depending on the specific stimuli, as suggested by Duncan and Humphreys (1989).

Further interesting examples of search asymmetry include cases for which neither target nor distractors have a primitive feature (such as color or orientation) that is absent in the other. Asymmetry is much weaker but still present. Figure 3 shows some typical examples. Although the saliencies of the more salient targets are only fractionally higher than the average feature saliency in rest of the image, this fraction is significant when the standard deviation $\sigma_s$ of the saliencies is small or when $z$ is large enough, thus making the search task easier.

## 3   Summary and Discussion

Early psychophysical studies (Treisman et al 1990) suggested that most aspects of visual search involve mechanisms of early vision. However, it has never been clear which visual areas or neural mechanisms might be responsible. To the best of my knowledge, this model is the first non-phenomenological model to understand the

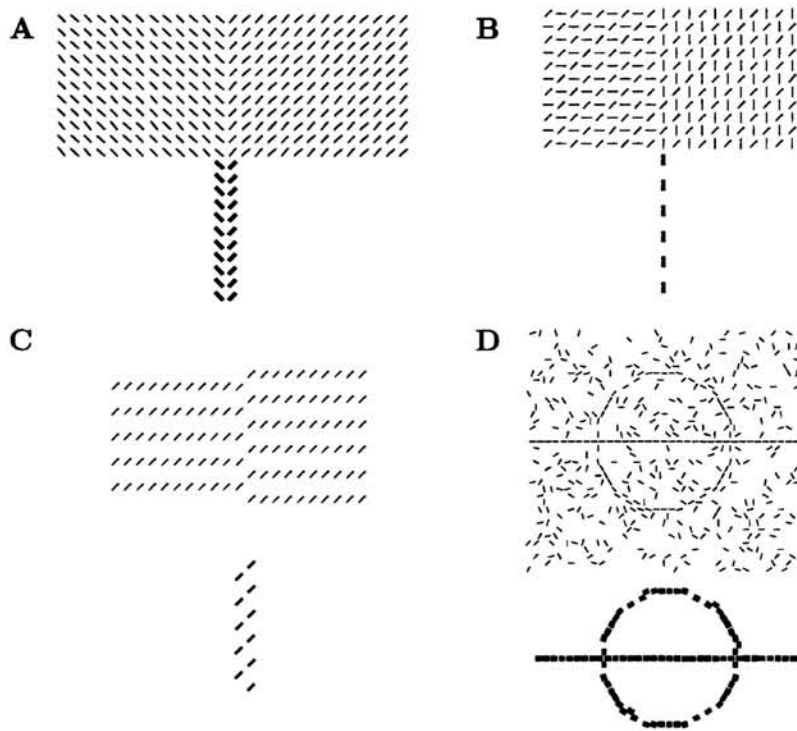

Figure 4: Four examples of model performance under various inputs. Each plots the visual input image at the top and the most activated bars in V1 cell outputs (using a threshold) at the bottom. Every visible bar in a given input image has the same input strength. A, B, and C demonstrate that the texture region boundaries have the highest output saliencies. D shows that the smooth contours are detected as the most salient against a background of noise.

neural bases of visual search phenomena (see Rubenstein and Sagi (1990) for a model of asymmetry using variances of the local image filter responses). This paper has shown that intra-cortical interactions in V1 can account for the qualitative phenomena of pop-out and asymmetry in visual search, assuming that the ease of detection is directly determined by the saliencies of targets. Of course, the task of search requires decision making and often visual attention, especially when the target does not spontaneously pop-out. The quantitative search times can only be modeled on the basis of an assumption of specific mechanisms for attention and decision making. Our model suggests, nevertheless, that pre-attentive V1 mechanisms play a significant and controlling role in such tasks. Furthermore, it suggests that some otherwise intractable phenomena can be understood without resorting to additional concepts such as textons (Julesz 1981) or defining certain image properties (such as closure and straightness) as having standard or reference values.

Our current implementation of V1 is still very simplistic. We have not yet included color, motion, or stereo inputs, nor multiscale sampling. Further, our input sampling density is very low. Consequently, the model cannot simulate many of the more complex input stimuli used in psychophysical experiments (Treisman and Gormican, 1988). An extended implementation is needed to test whether V1 mechanisms alone can qualitatively account for all or most types of search pop-out and asymmetries. Physiological evidence (Gilbert 1992) suggests that intracortical connections tend to link neurons with similar selectivities in other dimensions, such as color and stereo, in addition to orientation. This supports the idea that color, motion, and disparity are also primitive visual coding dimensions like orientation. We

believe that the example in Fig. 2A,B demonstrating pop-out versus serial search would be more convincing if color were included to simulate, for instance, a red 'X' among green 'X's with and without red 'O's in the background. Our current model does not explain why a slightly tilted line pops out more readily from vertical line distractors than the reverse. This is because our V1 model idealistically assumes rotational symmetry, and so vertical is not distinguished from other orientations. Neither our visual environment nor our visual system is in fact rotationally invariant.

The V1 model was originally proposed to account for pre-attentive contour enhancement and visual segmentation (Li 1998a, 1998b). The contextual influences mediated by the intracortical interactions enable each V1 neuron to process inputs from a local image area larger than its classical receptive field. This enables cortical neurons to detect image locations where translation invariance in the input image breaks down, and highlight these image locations with higher neural activities, making them conspicuous. These highlights mark candidate locations for image region (or object surface) boundaries, smooth contours and small figures against backgrounds, serving the purpose of pre-attentive segmentation. Fig. 4 demonstrates the performance of the model for pre-attentive segmentation. In each example, the visual inputs and the most salient outputs are shown. All examples are simulated using exactly the same model parameters as those used in examples of visual search. It is not too surprising that a model of pre-attentive segmentation in V1 can explain visual search phenomena. Indeed, pop out has been commonly understood as a sign of pre-attentive segmentation. Our model further suggests that asymmetry in visual search is partly a side-effect of pre-attentive segmentation. Our V1 model can in turn be improved using visual search as a diagnostic tool.

## Footnotes

[1]The terms 'edge' and 'bar' will be used interchangeably.

# References

[1] R. J. Douglas and K. A. Martin (1990) "Neocortex" in *Synaptic Organization of the Brain* ed. G. M. Shepherd. (Oxford University Press), 3rd Edition, pp389-438

[2] Duncan J. Humphreys G. *Psychological Review* 96: p1-26, (1989).

[3] C. D. Gilbert (1992) *Neuron.* **9**(1), 1-13.

[4] C. M. Gray and W. Singer (1989) *Proc. Natl. Acad. Sci. USA* **86**, 1698-1702.

[5] B. Julesz. (1981) *Nature* **290**, 91-97.

[6] M. K. Kapadia, M. Ito, C. D. Gilbert, and G. Westheimer (1995) *Neuron.* **15**(4), 843-56.

[7] J. J. Knierim and D. C. van Essen (1992) *J. Neurophysiol.* **67**, 961-980.

[8] Z. Li (1998a) in *Theoretical aspects of neural computation* Eds. Wong, K.Y.M, King, I, and D-Y Yeung, Springer-Verlag, 1998.

[9] Z. Li (1998b) *Neural Computation* 10(4) p 903-940.

[10] K.S. Rockland and J. S. Lund (1983) *J. Comp. Neurol.* **216**, 303-318

[11] Rubenstein B. and Sagi D. asymmetries" *J. Opt. Soc. Am. A* 9: 1632-1643 (1990).

[12] Treisman A, Cavanagh, P, Fischer B, Ramachandran V.S., and R. von der Heydt in *Visual perception, the Neurophysiological Foundations* Eds. L. Spillmann and J S. Werner, 1990 Academic Press.

[13] Treisman A. and Gormican S. (1988) *Psychological Rev.* **95**, 15-48.

[14] E. L. White (1989) *Cortical circuits* (Birkhauser).
